# Auction Mechanism Design for Multi-Robot Coordination

**Curt Bererton, Geoff Gordon, Sebastian Thrun, Pradeep Khosla**
{curt,ggordon,thrun,pkk}@cs.cmu.edu
Carnegie Mellon University
5000 Forbes Ave
Pittsburgh, PA 15217

## Abstract

The design of cooperative multi-robot systems is a highly active research area in robotics. Two lines of research in particular have generated interest: the solution of large, weakly coupled MDPs, and the design and implementation of market architectures. We propose a new algorithm which joins together these two lines of research. For a class of coupled MDPs, our algorithm automatically designs a market architecture which causes a decentralized multi-robot system to converge to a consistent policy. We can show that this policy is the same as the one which would be produced by a particular centralized planning algorithm. We demonstrate the new algorithm on three simulation examples: multi-robot towing, multi-robot path planning with a limited fuel resource, and coordinating behaviors in a game of paint ball.

## 1 Introduction

In recent years, the design of cooperative multi-robot systems has become a highly active research area within robotics [1, 2, 3, 4, 5, 6]. Many planning problems in robotics are best phrased as MDPs, defined over world states or—in case of partial observability—belief states [7]. However, existing MDP planning techniques generally scale poorly to multi-robot systems because of the curse of dimensionality: in general, it is exponentially harder to solve an MDP for $N$ agents than it is to solve a single-agent MDP, because the state and action space for $N$ robots can be exponentially larger than for a single-robot system. This enormous complexity has confined MDP planning techniques largely to single-robot systems.

In many cases, robots in a multi-robot system interact only in limited ways. Robots might seek not to collide with each other [1], coordinate their locations to carry out a joint task [4, 6], or consume a joint resource with limited availability [8, 9, 10]. While these problems are not trivially decomposed, they do not necessarily have the worst-case exponential complexity that characterizes the general case. However, so far we lack effective mechanisms for cooperatively solving such MDPs.

Handling this sort of limited interaction is exactly the strength of market-based planning algorithms [10, 12]: by focusing their attention on a limited set of important resources and ignoring all other interactions, these algorithms reduce the problem of cooperating with

other robots to the problem of deciding which resources to produce or consume. Market-based algorithms are particularly attractive for multi-robot planning because many common types of interactions can be phrased as constraints on resources such as space (two robots can't occupy the same location at once) and time (a robot can only work on a limited number of tasks at once).

From the point of view of these auction algorithms, the difficult part of the multi-robot planning problem is to compute the probability distribution of the price of each resource at every time step: the optimal price for a resource at time $t$ depends on how much each robot produces or consumes between now and time $t$, and what each robot's state is at time $t$. The resource usage and state depend on the robots' plans between now and time $t$, which in turn depend on the price. Worse yet, future resource usage depends on random events which can't be predicted exactly.

In this paper, we bring together resource-allocation tehniques from the auction and MDP literature. In particular, we propose a general technique for decomposing multi-robot MDP problems into "loosely coupled" MDPs which interact only through resource production and consumption constraints. The decomposition works by turning all interactions into streams of payments between robots, thereby allowing each robot to learn its own local value function. Prices can be attached to any function of the visitation frequencies of each robot's states and actions. The actual prices for these resources are set by a "master" agent; the master agent takes into account the possibility of re-allocating resources at each step, but it approximates the effect of interactions between robots.

Our approach generalizes a large body of previous literature in multi-robot systems, including prior work by Guestrin and Gordon [11]. Our algorithm can be distributed so that each robot reasons only about its own local interactions, and it always produces the same answer as a particular centralized planning algorithm.

## 2   MDPs, linear programs, and duals

A Markov Decision Process (MDP) is a tuple $\mathcal{M} = \{\mathcal{S}, \mathcal{A}, \mathcal{T}, c, \gamma, s_o\}$. $\mathcal{S}$ is a set of $N$ states. $\mathcal{A}$ is a set of $M$ actions. $\mathcal{T}$ is the dynamics $T(s', a, s) = p(s' \mid s, a)$. The reward function is $c : \mathcal{S} \times \mathcal{A} \mapsto \Re$. The discount factor is $\gamma \in [0, 1]$. Finally, $s_o \in \mathcal{S}$ is the initial state. For any MDP there is a *value function* which indicates how desirable any state is. It is defined as $V(s) = \max_a \left( c(s, a) + \gamma \sum_{s'} p(s' \mid s, a) V(s') \right)$. We can compute $V$ by solving the Bellman linear program (1). Once we have $V$, we can compute the optimal policy by one-step lookahead. Here $\mathbf{V} \in \Re^{\mathbf{N}}$ is the vector form of the value function. $\mathbf{c_a} \in \Re^{\mathbf{N}}$ is the immediate reward for taking action $a$ and $T_a \in \Re^{N \times N}$ is the matrix representation of the transition probabilities for action $a$. $\alpha$ is an arbitrary probability distribution over $\mathcal{S}$ which represents the probability of the MDP starting in a particular state. Typically, $\alpha$ is a vector in which one entry (the starting state) is set to one and all other entries are set to zero.

$$
\begin{array}{cc}
\min_{\mathbf{V}} \alpha \cdot \mathbf{V} & \\
\forall a : \ \mathbf{V} \geq \mathbf{c}_a + \gamma T_a \mathbf{V} & (1)
\end{array}
\qquad
\begin{array}{cc}
\max_{\mathbf{f}_a} \sum_a \mathbf{c}_a \cdot \mathbf{f}_a & \\
\sum_a \mathbf{f}_a - \gamma \sum_a T_a^T \mathbf{f}_a = \alpha & (2) \\
\forall a : \ \mathbf{f}_a \geq 0 &
\end{array}
$$

The dual of the Bellman LP gives us an interesting alternative from which to view the problem of finding an optimal policy. The dual of the Bellman LP is shown in (2). The vector $\mathbf{f_a}$ represents the expected number of times we perform action $a$ from each state.

For the remainder of the paper we will stack all of the $\mathbf{f}_a$ vectors into one large vector $\mathbf{f}$, and collect the equality constraints in (2) into $A\mathbf{f} = \mathbf{b}$. Subscripts (e.g., $\mathbf{f}_i$ or $A_i$) will distinguish the planning problems for different robots.

# 3 Algorithm

## 3.1 Loosely coupled MDPs

Our algorithm is designed for multi-robot problems that can be decomposed into separate single-robot MDPs which interact through the production or consumption of fictitious resources. These resources may be physical goods such as fuel; or they may be logical resources such as the right to pass over a bridge at a particular time, the right to explore an area of the environment, or the right to collect reward for achieving a particular subgoal. Time may be part of the individual robot states, in which case a resource could be the right to consume a unit of fuel at a particular time (a futures contract).

In more detail, each robot has a vector of state-action visitation frequencies $\mathbf{f}_i$ which must satisfy its own local dynamics $A_i \mathbf{f}_i = \mathbf{b}_i$. Its production or consumption of resources is defined by a matrix $C_i$: element $(j, k)$ of $C_i$ is the amount of resource $j$ which is produced or consumed by robot $i$ in state-action pair $k$. (So, $C_i \mathbf{f}_i$ is the vector of expected resource usages for robot $i$. The sign is arbitrary, so we will assume positive numbers correspond to consumption.) The robots interact through resource constraints: the instantaneous production and consumption of each resource must balance exactly.

This representation is in many ways related to an undirected dynamic Bayes network: each node of the network corresponds to the state and action of a single MDP, and a resource constraint involving a subset of the MDPs plays the role of a clique potential on the corresponding nodes. In this way it is similar to the representation of [11]; but, we do not assume any particular form for the $C_i$ matrices, while [11] effectively assumes that they are indicator functions of particular state or action variables.

In the same (trivial) sense as Bayes nets, our representation is completely general: by collapsing all robots into a single giant agent we can represent an arbitrary MDP. More importantly, in the more-typical case that some pieces of our model can be written as resource constraints, we can achieve an exponential savings in representation size compared to the monolithic planning problem.

## 3.2 Approximation

The resource constraints are what make loosely-coupled MDPs difficult to solve. They make the value of a joint state depend in a non-linear way on the states of the individual robots. However, by making a simple approximation we can remove the nonlinearity and so factor our planning problem: we relax the resource constraints so that they must only be satisfied in expectation over all time steps, rather than deterministically on each time step. Under this assumption, knowing the expected resources available to a robot allows that robot to plan independently: since $C_i \mathbf{f}_i$ is the vector of expected resource usages for robot $i$, adding the constraint $C_i \mathbf{f}_i = k$ to equation (2) gives us the single-robot resource-constrained planning problem.

The (approximate) global planning problem then becomes to determine an optimal resource allocation among robots and corresponding single-robot plans, or equivalently to determine the optimal resource prices and corresponding single-robot value functions. More formally, the planning problem is to solve (3):

$$
\begin{aligned}
& \max_{\mathbf{f}_i} \sum_i \mathbf{c}_i \cdot \mathbf{f}_i \\
& \forall i: \quad A_i \mathbf{f}_i = \mathbf{b}_i \\
(*) \quad & \qquad \sum_i C_i \mathbf{f}_i = \mathbf{d} \\
& \forall i: \quad \mathbf{f}_i \geq \mathbf{0}
\end{aligned}
\tag{3}
$$

Without the constraints marked $(*)$, this LP would represent a set of completely uncoupled robot planning problems. The constraints $(*)$ are the approximated resource constraints:

they say that expected production must equal expected consumption for each resource. The resource prices are the dual variables for $(*)$, and the local value functions are the dual variables for the remaining equality constraints.

The quality of our prices and value functions will depend on whether it is valid to assume a single price for each resource: if the prices stay constant then our approximate plan will translate perfectly to the physical world. On the other hand, if we are unlucky, we may find that prices are different than we had planned when we need to buy or sell. In this case our computed plan will contain overoptimistic, counterintuitive sequences of actions; for example, in the problem of section 3.4, two robots might each plan to break down at the same time and be towed by the other. The only way to fix this problem is to make a more accurate model; in the worst case we will have to combine several robots into one large MDP so that we can track their joint allocation of resources at all times.

### 3.3 Action selection

Because the value functions incorporate information about future actions and random events, the robots only need to look ahead a short time to choose good actions. So, the robots can run a simple auction to determine their best joint action: each individual robot estimates its future cost for each action by a single-step backup from its value function. The difference between these future costs then tells the robot how much it is willing to bid for the right to execute each action. The optimal joint action is then the feasible action with the highest sum of bids.

### 3.4 Example

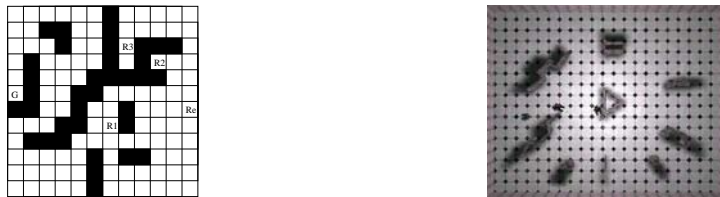

Figure 1: A simple example (left panel): the objective is to have all robots (R1,R2,R3) reach the goal (G) where they receive a reward. Any action may result in a robot becoming disabled, in which case it must be towed to the repair area (Re) to continue with the task. The grid shown here is significantly smaller than the problem solved in our experiments (right panel).

Figure 1 shows a simulator which contains 3 robots. Each robot receives a large reward upon reaching the goal but incurs a small cost for each step it takes. Robots can break whenever they take a step, but a functioning robot may tow a failed robot to the repair area and the repaired robot may then proceed to the goal. Each robot has the action set $\mathcal{A} = \{$ 8-connected move, pickup for towing, request tow$\}$. The state of each robot is its x position, its y position and its status {towing, going to goal, being towed, doing nothing}. If the grid is 300 by 300, then the state space size is $|S| = 300 \times 300 \times 4 = 360000$. The action space size is $|A| = 10$. The joint state space of all three robots is $|S_{joint}| = |S|^3$ and the joint action space is $|A| = 10^3$. Clearly, this problem size is such that ordinary MDP solution methods will be insufficient to determine the optimal value function.

However, this problem lends itself to resource-based decomposition because the robots only interact through towing. Specifically, we design our $C_i$ matrices to represent the constraint that the expected number of times a robot executes a pickup action at a position should be equal to the expected number of times some other robot executes a request-tow action. Thus, we have a weakly coupled MDP with robot interactions that can be modeled by linear constraints.

$\mathbf{p} \leftarrow 0 \quad G_i = [\,] \quad \Phi_i = [\,]$
**repeat**
  done $\leftarrow$ true
  **for** $i \leftarrow 1 \dots n$
    send prices $\mathbf{p}$ to robot $i$
    $\mathbf{f} \leftarrow$ frequencies from planning for robot $i$ with costs $\mathbf{c}_i - C_i^{\mathrm{T}}\mathbf{p}$
    send expected usage $\mathbf{g} = C_i\mathbf{f}$ and cost $\phi = \mathbf{c}_i \cdot \mathbf{f}$ to master
    **if** $\mathbf{g}$ is not already in $G_i$
      $G_i \leftarrow [G_i, \mathbf{g}] \quad \Phi_i \leftarrow [\Phi_i, \phi] \quad$ done $\leftarrow$ false
    **end if**
  **end for**
  $\mathbf{p} \leftarrow$ new dual variables from solving (4) with current $G_i$ and $\Phi_i$
**until** done

Figure 2: The decentralized planning algorithm based on Dantzig-Wolfe decomposition.

## 3.5 Dantzig-Wolfe decomposition

We have reduced the multi-robot planning problem to the problem of solving the LP (3). So, one possible planning algorithm is just to pass this LP to a pre-packaged linear-program solver. This planning algorithm can be fairly efficient, but it is completely centralized: each agent must communicate its entire dynamics to a central location and wait to receive its value function in return.

Instead of using this centralized algorithm, we want to produce the same outcome with a decentralized planner. To do so, we will apply Dantzig-Wolfe decomposition [13, chapter 24]. This decomposition splits our original LP (3) into a master LP (4) and one slave LP (5) for each robot $i$. It then solves each slave program repeatedly, generating a new value for $\mathbf{f}_i$ each time, and combines these solutions by inserting them into the master LP (Figure 2).

The Dantzig-Wolfe decomposition algorithm is guaranteed to terminate in a finite number of steps with the correct solution to our original LP and therefore with the correct local value functions. Each slave LP is the same as the corresponding robot's MDP except that it has different state-action costs; so, the robots can run standard MDP planners (which are often much faster than general LP solvers) to produce their plans. And, instead of sending whole MDPs and value functions back and forth, the Dantzig-Wolfe decomposition only needs to send resource prices and expected usages. The master program can be located on a separate agent, or on an arbitrary robot.

In more detail, the master and slave LPs are:

$$
(*) \quad
\begin{aligned}
\max_{\mathbf{q}_i} & \sum_i \mathbf{c}_i^{\mathrm{T}} F_i \mathbf{q}_i \\
& \sum_i C_i (F_i \mathbf{q}_i) = \mathbf{d} \\
& \forall i: \quad \mathbf{q}_i \geq 0 \\
& \forall i: \quad \sum_j q_{ij} = 1
\end{aligned}
\quad (4)
\qquad
\begin{aligned}
\max_{\mathbf{f}_i} & \ (\mathbf{c}_i^{\mathrm{T}} - \mathbf{p}^{\mathrm{T}} C_i)\mathbf{f}_i \\
& A_i \mathbf{f}_i = \mathbf{b}_i \\
& \mathbf{f}_i \geq \mathbf{0}
\end{aligned}
\quad (5)
$$

The master LP is the same as the original problem (3) except that $\mathbf{f}_i$ has been replaced by $F_i\mathbf{q}_i$. Each column of $F_i$ is one of the solutions $\mathbf{f}_i$ which we have computed for the $i$th slave LP. (For efficiency, instead of storing $F_i$ we keep $G_i = C_i F_i$ and $\Phi_i = \mathbf{c}_i^{\mathrm{T}} F_i$.) So, solving the master LP means finding a convex combination $\mathbf{q}_i$ of the known solutions for each slave LP. The slave LP is the same as a single-robot planning problem (2) except that its costs have been altered by subtracting $\mathbf{p}^{\mathrm{T}} C_i$. The vector $\mathbf{p}$ is the dual variable for the constraints $(*)$ from the last time we solved the master LP.

### 3.6 An economic interpretation

We have described how to use the Dantzig-Wolfe decomposition to derive an efficient distributed planning algorithm for loosely-coupled MDPs. In addition to being efficient and distributed, our algorithm has an intuitive economic interpretation which leads to interesting links with existing work on market architectures.

It is well known that the dual variables of a linear program have economic significance [14, 15]. Associated with each row of the constraint matrices $C_i$ in the master program (4) is a dual variable; that is, there is one dual variable $p_j$ for each resource $j$. We can interpret this dual variable as a price for resource $j$. To see why, notice that the slave program charges robot $i$ a cost of $p_j[C_i]_{j,k}$ each time it visits state-action pair $k$, and that visiting state-action pair $k$ consumes an amount $[C_i]_{j,k}$ of resource $j$.

The Dantzig-Wolfe algorithm can be interpreted as a search for optimal resource prices. The master agent repeatedly asks the robots what they would do if the prices were $\mathbf{p}$, then tries to combine their answers to produce a good plan for all the robots together. As it combines the single-robot plans, it notices whether it could achieve a higher reward by increasing or decreasing the supply of each resource; if there is an undersupply of a resource the master agent assigns it a high price, and if there is an oversupply the master agent assigns it a low price.

## 4 Experimental results

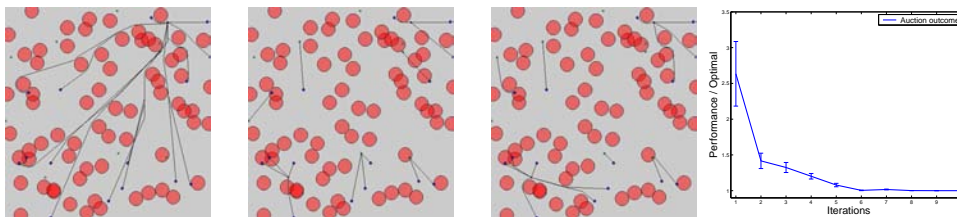

Figure 3: Auctions for multi-robot path planning with limited fuel usage. Left to right: in an auction based on the assumption of cheap fuel, all robots go to the globally most tempting goal. If we assume very expensive fuel, each robot crashes through obstacles and goes to its closest goal. With the optimal fuel price, the auction trades goal quality against distance to achieve the best possible total cost. As our algorithm learns better prices, the auction's outcomes approach the optimal policy.

Our experiments are divided into two groups. First, to investigate the convergence rate of our algorithm, we collected data from multiple runs on randomly-generated synthetic problems. Second, to investigate scaling, we applied the algorithm to a large, realistic problem taken from our ongoing research into robotic laser tag [16].

In our synthetic problem, we randomly place circular obstacles inside a bounded arena to create a maze. We then place 15 robots in random starting locations and ask them to plan paths to 10 random goals. Each robot can choose whichever goal it wants, but must pay a random goal-specific price. The robots are coupled through a constraint on fuel usage: there is a quadratic penalty on total path length.

In this problem, our algorithm starts from an arbitrary initial guess at the value of a unit of fuel (which causes the individual robots to make poor policy decisions) and rapidly improves the estimated value by examining the individual robot plans. We averaged the performance of our algorithm on 20 random instances; the results are shown in Figure 3.

To demonstrate scaling, we used our learning algorithm to coordinate the robot towing problem in the simulation shown in figure 4, with a grid size of $300 \times 300$ and 9 robots. Many more robots could be handled, but because we only coordinated towing and not path

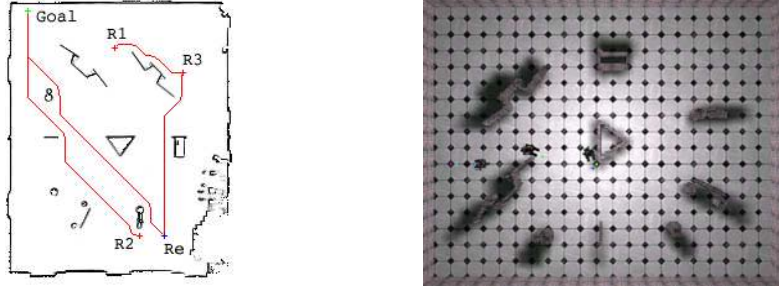

Figure 4: Left: an example of the output of the algorithm on a towing problem on a map generated using the robots on the right. Note that the nearest live robot (R1) tows the damaged robot to the repair area before heading to the goal. This type of problem was solved for up to 9 robots. Right: Multi-robot paint ball simulator.

planning in this example, there was a bottleneck at the repair area due to the unmodeled coordination. The resulting paths executed in a sample problem are shown in figure 4. Because our algorithm uses an arbitrary MDP planner as a subroutine, very large problems can be solved by combining our approach with fast planning algorithms.

Figure 4 shows the simulator in which we applied the method to multi-robot paint ball. The rules of the game are that the last team standing wins and that it takes 4 hits to cause a robot to fail. There is a repair area to which a tagged teammate may be towed in order to repair it so that it may continue to play. Robots can only see each other when there are no obstacles between them.

In this problem, we use our method to select and coordinate predefined policies. Policies used are: do nothing, attack target $i$, coordinated attack (with a teammate) target $i$, tow teammate $i$, and be repaired. Currently these policies are hand specified, but in future work we would like to apply policy search methods to learn these policies. The objective of our multi-robot planner is to determine at a given time which fixed policy each robot on the team should be executing so that the team will perform better. Coordination constraints are that any coordinated attacks or towing/repairing must be consistent: if teammate 1 requests a tow from teammate 2, then teammate 2 must perform a tow of teammate 1.

To solve the slave problems, we use rollouts of the given policies. This allows us to handle partial observability as each enemy is tracked with a particle filter, and the particle filter distribution is used when performing rollouts. Enemy positions are sampled from the particle filters at the beginning of each rollout and each policy is evaluated over several possible enemy position combinations to determine the performance of a policy. The robots replan at fixed intervals; the simulation is halted while planning occurs.

We compared our coordination planner to a similar planner without coordination. Each planner was played against a default behavior of "attack nearest enemy" over 50 games. The uncoordinated planner won 42 of 50 games over the default behavior. The coordinated planner won 48 of 50 games against the default behavior. Thus, the addition of coordination (via our factored planning algorithm) significantly improved the performance.

## 5   Conclusions

We have developed a decentralized method for solving large loosely-coupled multi-robot planning problems. Our algorithm works by finding an optimal solution to an approximate planning problem in which resource constraints hold only in expectation. It has an intuitive economic interpretation which facilitates its application to new problems. And, it can be combined with previous MDP decomposition methods, allowing the user to mix and match

which methods are best suited to their problem. We have applied our algorithm to multi-robot towing, optimal use of fuel in a multi-robot path planning problem, and planning for multi-robot paintball.

## Acknowledgements

This project was supported by DARPA's MICA and MARS programs.

## References

[1] M. Bennewitz, W. Burgard, and S. Thrun. Optimizing schedules for prioritized path planning of multi-robot systems. In *IEEE International Conference on Robotics and Automation (ICRA)*, Seoul, Korea, 2001. ICRA.

[2] Cao Y.U., Fukunaga A.S., and Kahng A.B. Cooperative mobile robotics: Antecedents and directions. *Autonomous Robots*, 4:1–23, 1997.

[3] D. Goldberg and M.J. Matarić. Robust behavior-based control for distributed multi-robot collection tasks. Technical Report IRIS-00-387, USC Institute for Robotics and Intelligent Systems, 2000.

[4] H. Kitano, editor. *Proceedings of RoboCup-97: The First Robot World Cup Soccer Games and Conferences*, Berlin, 1998. Springer Verlag.

[5] S.I. Roumeliotis and G.A Bekey. Distributed multi-robot localization. In *Proceedings of the International Symposium on Distributed Autonomous Robotic Systems (DARS 2000)*, pages 179–188, Knoxville, Tenneessee, 2000.

[6] J. Salido, J. Dolan, J. Hampshire, and P.K. Khosla. A modified reactive control framework for cooperative mobile robots. In *Proceedings of the International Conference on Sensor Fusion and Decentralized Control*, pages 90–100, Pittsburgh, PA, 1997. SPIE.

[7] L.P. Kaelbling, M.L. Littman, and A.R. Cassandra. Planning and acting in partially observable stochastic domains. *Artificial Intelligence*, 101(1-2):99–134, 1998.

[8] W. Burgard, D. Fox, M. Moors, R. Simmons, and S. Thrun. Collaborative multi-robot exploration. In *Proceedings of the IEEE International Conference on Robotics and Automation (ICRA)*, San Francisco, CA, 2000. IEEE.

[9] L. E. Parker. On the design of behavior-based multi-robot teams. *Journal of Advanced Robotics*, 10(6), 1996.

[10] R. Zlot, A. Stentz, M. Dias, and S. Thayer. Multi-robot exploration controlled by a market economy, 2002.

[11] Carlos Guestrin and Geoffrey Gordon. Distributed planning in hierarchical factored MDPs. In A. Darwiche and N. Friedman, editors, *Uncertainty in Artificial Intelligence (UAI)*, volume 18, 2002.

[12] Brian P. Gerkey and Maja J Mataric. Sold!: Market methods for multi-robot control.

[13] George B. Dantzig. *Linear Programming and Extensions*. Princeton University Press, 1963.

[14] Ronald Rardin. *Optimization in Operations Research*. Prentice Hall, 1998.

[15] Vasek Chvatal. *Linear Programming*. W.H. Freeman and Company, 1983.

[16] Matthew Rosencrantz, Geoffrey Gordon, and Sebastian Thrun. Locating moving entities in dynamic indoor environments. In *ACM AGENTS*, 2003.

[17] M. Dias and A. Stentz. A market approach to multirobot coordination, 2001.
